# Direction Selective Silicon Retina
# that uses Null Inhibition

**Ronald G. Benson and Tobi Delbrück**
Computation and Neural Systems Program, 139-74
California Institute of Technology
Pasadena CA 91125
email: benson@cns.caltech.edu and tdelbruck@caltech.edu

## Abstract

Biological retinas extract spatial and temporal features in an attempt to reduce the complexity of performing visual tasks. We have built and tested a silicon retina which encodes several useful temporal features found in vertebrate retinas. The cells in our silicon retina are selective to direction, highly sensitive to positive contrast changes around an ambient light level, and tuned to a particular velocity. Inhibitory connections in the null direction perform the direction selectivity we desire. This silicon retina is on a $4.6 \times 6.8mm$ die and consists of a $47 \times 41$ array of photoreceptors.

## 1    INTRODUCTION

The ability to sense motion in the visual world is essential to survival in animals. Visual motion processing is indispensable; it tells us about predators and prey, our own motion and image stablization on the retina. Many algorithms for performing early visual motion processing have been proposed [HK87] [Nak85]. A key salient feature of motion is direction selectivity, *ie* the ability to detect the direction of moving features. We have implemented Barlow and Levick's model, [BHL64], which hypothesizes inhibition in the null direction to accomplish direction selectivity.

In contrast to our work, Boahen, [BA91], in these proceedings, describes a silicon retina that is specialized to do spatial filtering of the image. Mahowald, [Mah91], describes a silicon retina that has surround interactions and adapts over mulitple time scales. Her silicon retina is designed to act as an analog preprocessor and

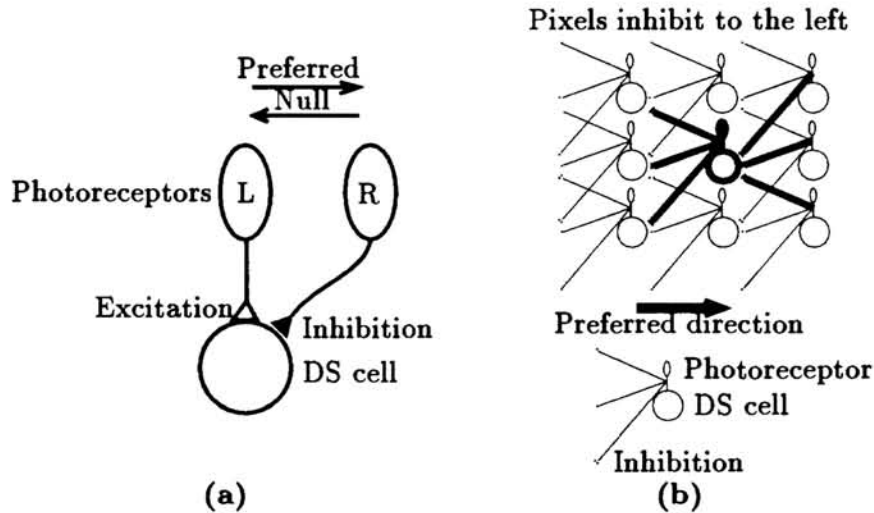

Figure 1: **Barlow and Levick model of direction selectivity (DS). (a) Shows how two cells are connected in an inhibitory fashion and (b) a mosaic of such cells.**

so the gain of the output stage is rather low. In addition there is no rectification into on- and off-pathways. This and earlier work on silicon early vision systems have stressed spatial processing performed by biological retinas at the expense of temporal processing.

The work we describe here and the work described by Delbrück, [DM91], emphasizes temporal processing. Temporal differentiation and separation of intensity changes into on- and off-pathways are important computations performed by vertebrate retinas. Additionally, specialized vertebrate retinas, [BHL64], have cells which are sensitive to moving stimuli and respond maximally to a preferred direction; they have almost zero response in the opposite or null direction. We have designed and tested a silicon retina that models these direction selective velocity tuned cells. These receptors excite cells which respond to positive contrast changes only and are selective for a particular direction of stimuli. Our silicon retina may be useful as a preprocessor for later visual processing and certainly as an enhancement for the already existing spatial retinas. It is a striking demonstration of the perceptual saliency of contrast changes and directed motion in the visual world.

## 2    INHIBITION IN THE NULL DIRECTION

Barlow and Levick, [BHL64], described a mechanism for direction selectivity found in the rabbit retina which postulates inhibitory connections to achieve the desired direction selectivity. Their model is shown in Figure 1(a) . As a moving edge passes over the photoreceptors from left to right, the left photoreceptor is excited first, causing its direction selective (DS) cell to fire. The right photoreceptor fires when the edge reaches it and since it has an inhibitory connection to the left DS cell, the right photoreceptor retards further output from the left DS cell. If an edge is moving in the opposite or null direction (right to left), the activity evoked in the right photoreceptor completely inhibits the left DS cell from firing, thus creating a direction selective cell.

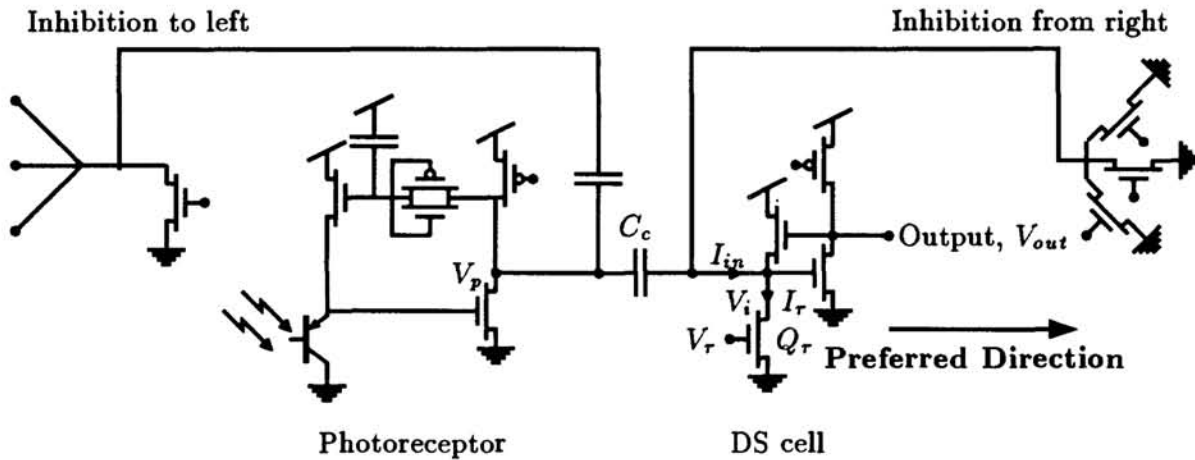

**Figure 2: Photoreceptor and direction selective (DS) cell. The output of the high-gain, adaptive photoreceptor is fed capacitively to the input of the DS cell. The output of the photoreceptor sends inhibition to the left. Inhibition from the right photoreceptors connect to the input of the DS cell.**

In the above explanation with the edge moving in the preferred direction (left to right), as the edge moves faster, the inhibition from leading photoreceptors truncates the output of the DS cell ever sooner. In fact, it is this inhibitory connection which leads to velocity tuning in the preferred direction.

By tiling these cells as shown in Figure 1(b), it is possible to obtain an array of directionally tuned cells. This is the architecture we used in our chip. Direction selectivity is inherent in the connections of the mosaic, *ie* the hardwiring of the inhibitory connections leads to directionally tuned cells.

## 3   PIXEL OPERATION

A pixel consists of a photoreceptor, a direction selective (DS) cell and inhibition to and from other pixels as shown in Figure 2. The photoreceptor has high-gain and is adaptive [Mah91, DM91]. The output from this receptor, $V_p$, is coupled into the DS cell which acts as a rectifying gain element, [MS91], that is only sensitive to positive-going transitions due to increases in light intensity at the receptor input. Additionally, the output from the photoreceptor is capacitively coupled to the inhibitory synapses which send their inhibition to the left and are coupled into the DS cell of the neighboring cells.

A more detailed analysis of the DS cell yields several insights into this cell's functionality. A step increase of $\Delta V$ at $V_p$, caused by a step increase in light intensity incident upon the phototransistor, results in a charge injection of $C_c \Delta V$ at $V_i$. This charge is leaked away by $Q_\tau$ at a rate $I_\tau$, set by voltage $V_\tau$. Hence, to first order, the output pulse width $T$ is simply

$$T = \frac{C_c \Delta V}{I_\tau}.$$

There is also a threshold minimum step input size that will result in enough change

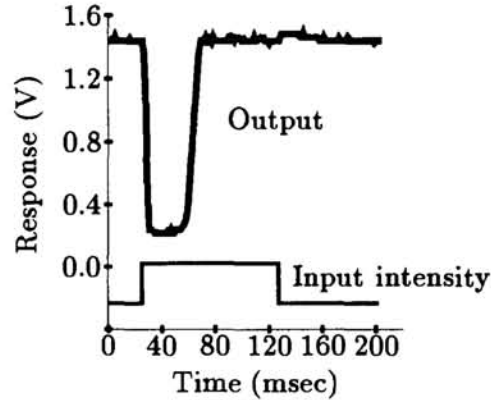

**Figure 3: Pixel response to intensity step. Bottom trace is intensity; top trace is pixel output.**

in $V_i$ to pull $V_{\text{out}}$ all the way to ground. This threshold is set by $C_c$ and the gain of the photoreceptor.

When the input to the rectifying gain element is not a step, but instead a steady increase in voltage, the current $I_{\text{in}}$ flowing into node $V_i$ is

$$I_{\text{in}} = C_c \dot{V}_p.$$

When this current exceeds $I_\tau$ there is a net increase in the voltage $V_i$, and the output $V_{\text{out}}$ will quickly go low. The condition $I_{\text{in}} = I_\tau$ defines the threshold limit for stimuli detection, *i.e.* input stimuli resulting in an $I_{\text{in}} < I_\tau$ are not perceptible to the pixel. For a changing intensity $\dot{I}$, the adaptive photoreceptor stage outputs a voltage $V_p$ proportional to $\dot{I}/I$, where $I$ is the input light intensity. This photoreceptor behavior means that the pixel threshold will occur at whatever $\dot{I}/I$ causes $C_c \dot{V}_p$ to exceed the constant current $I_\tau$.

The inhibitory synapses (shown as Inhibition from right in Figure 2) provide additional leakage from $V_i$ resulting in a shortened response width from the DS cell.

This analysis suggests that a characterization of the pixel should investigate both the response amplitude, measured as pulse width versus input intensity step size, and the response threshold, measured with temporal intensity contrast. In the next section we show such measurements.

# 4   CHARACTERIZATION OF THE PIXEL

We have tested both an isolated pixel and a complete 2-dimensional retina of $47 \times 41$ pixels. Both circuits were fabricated in a $2\mu$m p-well CMOS double poly process available through the MOSIS facility. The retina is scanned out onto a monitor using a completely integrated on-chip scanner[MD91]. The only external components are a video amplifier and a crystal.

We show a typical response of the isolated pixel to an input step of intensity in Figure 3. In response to the input step increase of intensity, the pixel output goes low and saturates for a time set by the bias $V_\tau$ in Figure 2. Eventually the pixel recovers and the output returns to its quiescent level. In response to the step decrease of intensity there is almost no response as seen in Figure 3.

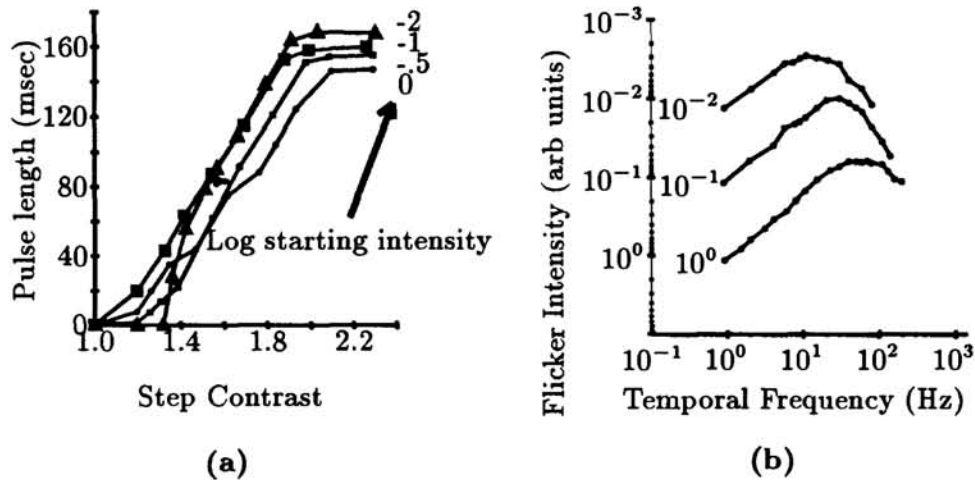

**Figure 4: (a) Pulse width of response as function of input contrast step size.** The abscissa is measured in units of ratio-intensity, i.e., a value of 1 means no intensity step, a value of 1.1 means a step from a normalized intensity of 1 to a normalized intensity of 1.1, and so forth. The different curves show the response at different absolute light levels; the number in the figure legend is the log of the absolute intensity. **(b) Receptor threshold measurements.** At each temporal frequency, we determined the minimum necessary amplitude of triangular intensity variations to make the pixel respond. The different curves were taken at different background intensity levels, shown to the left of each curve. For example, the bottom curve was taken at a background level of 1 unit of intensity; at 8 Hz, the threshold occurred at a variation of 0.2 units of intensity.

The output from the pixel is essentially quantized in amplitude, but the resulting pulse has a finite duration related to the input intensity step. The analysis in Section 3 showed that the output pulse width, $T$, should be linear in the input intensity contrast step. In Figure 4(a), we show the measured pulse-width as a function of input contrast step. To show the adaptive nature of the receptor, we did this same measurement at several different absolute intensity levels.

Our silicon retina sees some features of a moving image and not others. Detection of a moving feature depends on its contrast and velocity. To characterize this behavior, we measured a receptor's thresholds for intensity variations, as a function of temporal frequency.

These measurements are shown in Figure 4(b); the curves define "zones of visibility"; if stimuli lie below a curve, they are visible, if they fall above a curve they are not. (The different curves are for different absolute intensity levels.) For low temporal frequencies stimuli are visible only if they are high contrast; at higher temporal frequencies, but still below the photoreceptor cutoff frequency, lower contrast stimuli are visible. Simply put, if the input image has low contrast and is slowly moving, it is not seen. Only high contrast or quickly moving features are salient stimuli. More precisely, for temporal frequencies below the photoreceptor cutoff frequency, the threshold occurs at a constant value of the temporal intensity contrast $\dot{I}/I$.

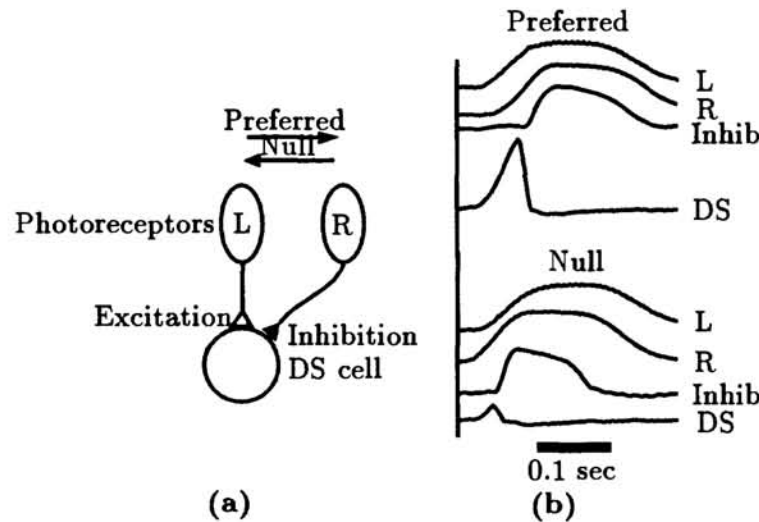

(a)                                        (b)

**Figure 5:** (a) shows the basic connectivity of the tested cell. (b) top trace is the response due to an edge moving in the preferred direction (left to right). (b) bottom trace is the response due to an edged moving in the null direction (right to left).

## 5  NULL DIRECTION INHIBITION PROPERTIES

We performed a series of tests to characterize the inhibition for various orientations and velocities. The data in Figure 5(b) shows the outputs of two photoreceptors, the inhibitory signal and the output of a DS cell. The top panel in Figure 5(b) shows the outputs in the preferred direction and the bottom panel shows them in the null direction. Notice that the output of the left photoreceptor (L in Figure 5(b) top panel) precedes the right (R). The output of the DS cell is quite pronounced, but is truncated by the inhibition from the right photoreceptor. On the other hand, the bottom panel shows that the output of the DS cell is almost completely truncated by the inhibitory input.

A DS cell receives most inhibition when the stimulus is travelling exactly in the null direction. As seen in Figure 6(a) as the angle of stimulus is rotated, the maximum response from the DS cell is obtained when the stimulus is moving in the preferred direction (directly opposite to the null direction). As the bar is rotated toward the null direction, the response of the cell is reduced due to the increasing amount of inhibition received from the neighboring photoreceptors.

If a bar is moving in the preferred direction with varying velocity, there is a velocity, $V_{max}$, for which the DS cell responds maximally as shown in Figure 6(b). As the bar is moved faster than $V_{max}$, inhibition arrives at the cell sooner, thus truncating the response. As the cell is moved slower than $V_{max}$, less input is provided to the DS cell as described in Section 3. In the null direction (negative in Figure 6(b)) the cell does not respond, as expected, until the bar is travelling fast enough to beat the inhibition's onset (recall delay from Figure 5).

In Figure 7 we show the response of the entire silicon retina to a rotating fan. When the fan blades are moving to the left the retina does not respond, but when moving to the right, note the large response. Note the largest response when the blades are moving exactly in the preferred direction.

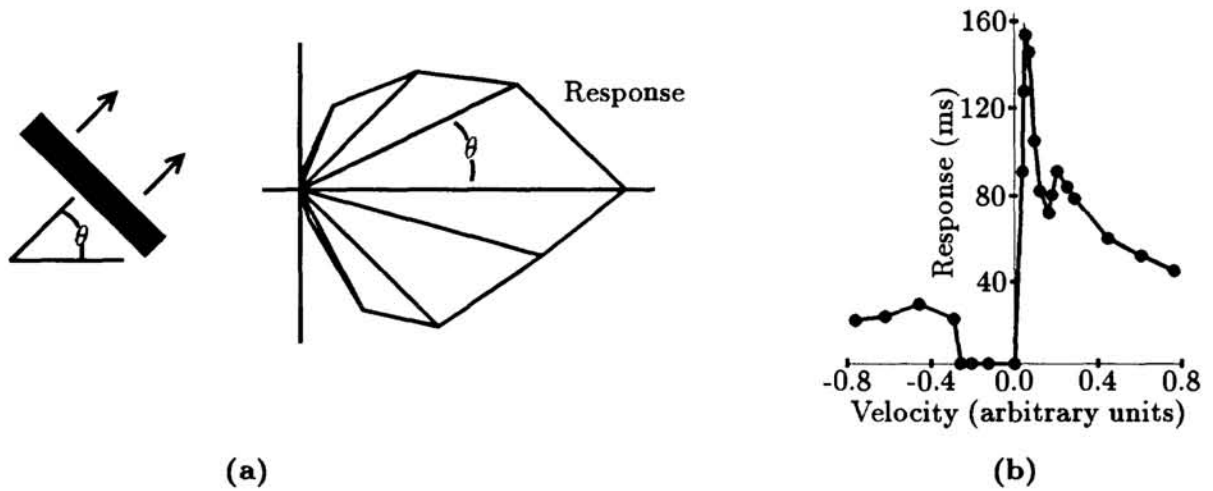

(a)                                                                 (b)

Figure 6:  (a) polar plot which shows the pixels are directionally tuned.
(b) shows velocity tuning of the DS cell (positive velocities are in the pre-
ferred direction).

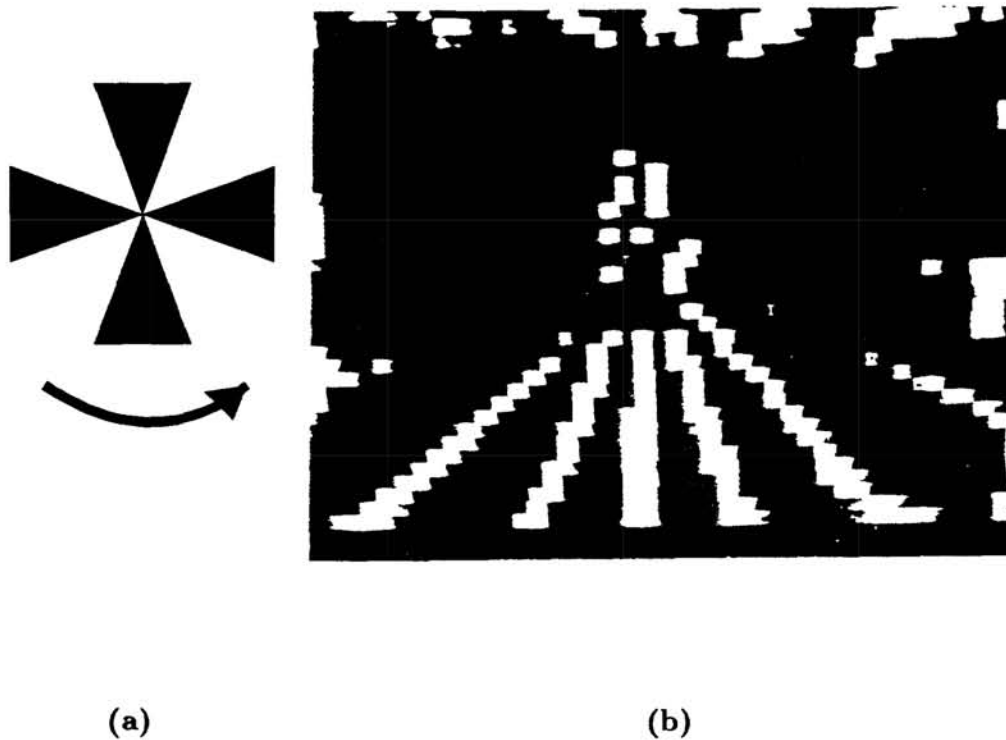

(a)                                                 (b)

Figure 7: (a)  Rotating fan used as stimulus to the retina. (b)  Output of the
retina.

# 6  CONCLUSION

We have designed and tested a silicon retina that detects temporal changes in an image. The salient image features are sufficiently high contrast stimuli, relatively fast increase in intensity (measured with respect to the recent past history of the intensity), direction and velocity of moving stimuli. These saliency measures result in a large compression of information, which will be useful in later processing stages.

## Acknowledgments

Our thanks to Carver Mead and John Hopfield for their guidance and encouragement, to the Office of Naval Research for their support under grant NAV N00014-89-J-1675, and, of course, to the MOSIS fabrication service.

# References

[BA91]   K. Boahen and A. Andreou. A contrast sensitive silicon retina with reciprocal synapses. In S. Hanson J. Moody and R. Lippmann, editors, *Advances in Neural Information Processing Systems, Volume 4*. Morgan Kaufmann, Palo Alto, CA, 1991.

[BHL64]   H.B. Barlow, M.R. Hill, and W.R. Levick. Retinal ganglion cells responding selectively to direction and speed of image motion in the rabbit. *J. Physiol.*, 173:377–407, 1964.

[DM91]   T. Delbrück and Carver Mead. Silicon adaptive photoreceptor array that computes temporal intensity derivatives. In *Proc. SPIE 1541*, volume 1541-12, pages 92–99, San Diego, CA, July 1991. Infrared Sensors: Detectors, Electronics, and Signal Processing.

[HK87]   E. Hildreth and C. Koch. The analysis of visual motion: From computational theory to neuronal mechanisms. *Annual Review in Neuroscience*, 10:477–533, 1987.

[Mah91]   M.A. Mahowald. Silicon retina with adaptive photoreceptor. In *SPIE Technical Symposia on Optical Engineering and Photonics in Aerospace Sensing*, Orlando, FL, April 1991. Visual Information Processing: From Neurons to Chips.

[MD91]   C.A. Mead and T. Delbrück. Scanners for use in visualizing analog VLSI circuitry. *Analog Integrated Circuits and Signal Processing*, 1:93–106, 1991.

[MS91]   C.A. Mead and R. Sarpeshkar. An axon circuit. Internal Memo, Physics of Computation Laboratory, Caltech, 1991.

[Nak85]   K. Nakayama. Biological image motion processing: A review. *Vision Research*, 25(5):625–660, 1985.
